# Second Order Bilinear Discriminant Analysis for single-trial EEG analysis

**Christoforos Christoforou**
Department of Computer Science
The Graduate Center of the City University of New York
365 Fifth Avenue
New York, NY 10016-4309
cchristoforou@gc.cuny.edu

**Paul Sajda**
Department of Biomedical Engineering
Columbia University
351 Engineering Terrace Building, MC 8904
1210 Amsterdam Avenue
New York, NY 10027
ps629@columbia.edu

**Lucas C. Parra**
Department of Biomedical Engineering
The City College of The City University of New York
Convent Avenue 138th Street
New York,NY 10031, USA
parra@ccny.cuny.edu

## Abstract

Traditional analysis methods for single-trial classification of electro-encephalography (EEG) focus on two types of paradigms: phase locked methods, in which the amplitude of the signal is used as the feature for classification, e.g. event related potentials; and second order methods, in which the feature of interest is the power of the signal, e.g. event related (de)synchronization. The procedure for deciding which paradigm to use is *ad hoc* and is typically driven by knowledge of the underlying neurophysiology. Here we propose a principled method, based on a bilinear model, in which the algorithm simultaneously learns the best first and second order spatial and temporal features for classification of EEG. The method is demonstrated on simulated data as well as on EEG taken from a benchmark data used to test classification algorithms for brain computer interfaces.

## 1 Introduction

### 1.1 Utility of discriminant analysis in EEG

Brain computer interface (BCI) algorithms [1][2][3][4] aim to decode brain activity, on a single-trial basis, in order to provide a direct control pathway between a user's intentions and a computer. Such an interface could provide "locked in patients" a more direct and natural control over a neuroprosthesis or other computer applications [2]. Further, by providing an additional communication

channel for healthy individuals, BCI systems can be used to increase productivity and efficiency in high-throughput tasks [5, 6].

Single-trial discriminant analysis has also been used as a research tool to study the neural correlates of behavior. By extracting activity that differs maximally between two experimental conditions, the typically low signal-noise ratio of EEG can be overcome. The resulting discriminant components can be used to identify the spatial origin and time course of stimulus/response specific activity, while the improved SNR can be leveraged to correlate variability of neural activity across trials to behavioral variability and behavioral performance [7, 5]. In essence, discriminant analysis adds to the existing set of multi-variate statistical tools commonly used in neuroscience research (ANOVA, Hoteling $T^2$, Wilks' $\Lambda$ test).

## 1.2    Linear and quadratic approaches

In EEG the signal-to-noise ratio of individual channels is low, often at -20dB or less. To overcome this limitation, all analysis methods perform some form of averaging, either across repeated trials, across time, or across electrodes. Traditional EEG analysis averages signals across many repeated trials for individual electrodes. A conventional method is to average the measured potentials following stimulus presentation, thereby canceling uncorrelated noise that is not reproducible from one trial to the next. This averaged activity, called an event related potential (ERP), captures activity that is time-locked to the stimulus presentation but cancels evoked oscillatory activity that is not locked in phase to the timing of the stimulus. Alternatively, many studies compute the oscillatory activity in specific frequency bands by filtering and squaring the signal prior to averaging. Thus, changes in oscillatory activity are termed event related synchronization or desynchronization (ERS/ERD).

Surprisingly, discriminant analysis methods developed thus far by the machine learning community have followed this dichotomy: First order methods in which the amplitude of the EEG signal is considered to be the feature of interest in classification – corresponding to ERP – and second order methods in which the power of the feature is considered to be of importance for classification – corresponding to ERS/ERD. First order methods include temporal filtering + thresholding [2], hierarchical linear classifiers [5] and bilinear discriminant analysis [8, 9]. Second order methods include the logistic regression with a quadratic term [11] and the well known common spatial patterns method (CSP) [10] and its variants: common spatio-spectral patterns (CSSP)[12], and common sparse spectral spatial patterns (CSSSP)[13] .

Choosing what kind of features to use traditionally has been an *ad hoc* process motivated by knowledge of the underlying neurophysiology and task. From a machine-learning point of view, it seems limiting to commit *a priori* to only one type of feature. Instead it would be desirable for the analysis method to extract the relevant neurophysiological activity *de novo* with minimal prior expectations. In this paper we present a new framework that combines both the first order features and the second order features in the analysis of EEG. We use a bilinear formulation which can simultaneously extract spatial linear components as well as temporal (filtered) features.

## 2    Second order bilinear discriminant analysis

### 2.1    Problem setting

Given a set of sample points $\mathcal{D} = \{\mathbf{X}_n, y_n\}_{n=1}^N, \mathbf{X} \in \mathbb{R}^{D \times T}, y \in \{-1, 1\}$ , where $\mathbf{X}_n$ corresponds to the EEG signal of $D$ channels and $T$ sample points and $y_n$ indicate the class that corresponds to one of two conditions (e.g. right or left hand imaginary movement, stimulus versus control conditions, etc.), the task is then to predict the class label $y$ for an unobserved trial $\mathbf{X}$.

### 2.2    Second order bilinear model

Define a function,

$$f(\mathbf{X}; \theta) = C \operatorname{Trace}(\mathbf{U}^{\mathrm{T}} \mathbf{X} \mathbf{V}) + (1 - C) \operatorname{Trace}(\mathbf{\Lambda} \mathbf{A}^{\mathrm{T}} (\mathbf{X} \mathbf{B})(\mathbf{X} \mathbf{B})^{\mathrm{T}} \mathbf{A}) \qquad (1)$$

where $\theta = \{\mathbf{U} \in \mathbb{R}^{D \times R}, \mathbf{V} \in \mathbb{R}^{T \times R}, \mathbf{A} \in \mathbb{R}^{D \times K} \; \mathbf{B} \in \mathbb{R}^{T \times T'}\}$ are the parameters of the model, $\mathbf{\Lambda} \in diag(\{-1, 1\})$ a given diagonal matrix with elements $\{-1, 1\}$ and $C \in [0, 1]$. We consider the

following *discriminative model*; we model the log-odds ratio of the posterior class probability to be the sum of a bilinear function with respect to the EEG signal amplitude and linear with respect to the second order statistics of the EEG signal:

$$\log \frac{P(y=+1|\mathbf{X})}{P(y=-1|\mathbf{X})} = f(\mathbf{X}|\theta) \qquad (2)$$

### 2.2.1 Interpretation of the model

The first term of the equation (1) can be interpreted as a spatio-temporal projection of the signal, under the bilinear model, and captures the first order statistics of the signal. Specifically, the columns $\mathbf{u}_r$ of $\mathbf{U}$ represent $R$ linear projections in space (rows of $\mathbf{X}$). Similarly, each of the $R$ columns of $\mathbf{v}_k$ in matrix $\mathbf{V}$ represent linear projections in time (columns of $\mathbf{X}$). By re-writing the term as:

$$\text{Trace}(\mathbf{U}^{\mathrm{T}}\mathbf{X}\mathbf{V}) = \text{Trace}(\mathbf{V}\mathbf{U}^{\mathrm{T}}\mathbf{X}) = \text{Trace}(\mathbf{W}^{\mathrm{T}}\mathbf{X}) \qquad (3)$$

where we defined $\mathbf{W} = \mathbf{U}\mathbf{V}^{\mathrm{T}}$, it is easy to see that the bilinear projection is a linear combination of elements of $\mathbf{X}$ with the $rank - R$ constrained on $\mathbf{W}$. This expression is linear in $\mathbf{X}$ and thus captures directly the amplitude of the signal directly. In particular, the polarity of the signal (positive evoked response versus negative evoked response) will contribute significantly to discrimination if it is consistent across trials. This term, therefore, captures phase locked event related potentials in the EEG signal.

The second term of equation (1), is a projection of the power of the filtered signal, which captures the second order statistics of the signal. As before, each column of matrix $\mathbf{A}$ and $\mathbf{B}$, represent components that project the data in space and time respectively. Depending on the structure one enforces in matrix $\mathbf{B}$ different interpretations of the model can be archived. In the general case where no structure on $\mathbf{B}$ is assumed, the model captures a linear combination of the elements of a $rank - T'$ second order matrix approximation of the signal $\Sigma = \mathbf{X}\mathbf{B}(\mathbf{X}\mathbf{B})^{\mathrm{T}}$. In the case where Toeplitz structure is enforced on $\mathbf{B}$, then $\mathbf{B}$ defines a temporal filter on the signal and the model captures the linear combination of the power of the second order matrix of the filtered signal. For example if $\mathbf{B}$ is fixed to a Toeplitz matrix with coefficients corresponding to a 8Hz-12Hz band pass filter, then the second term is able to extract differences in the alpha-band which is known to be modulated during motor related tasks. Further, by learning $\mathbf{B}$ from the data, we may be able to identify new frequency bands that have so far not been identified in novel experimental paradigms. The spatial weights $\mathbf{A}$ together with the $\text{Trace}$ operation ensure that the power is measured, not in individual electrodes, but in some component space that may reflect activity distributed across several electrodes.

Finally, the scaling factor $\lambda$ (which may seem superfluous given the available degrees of freedom) is necessary once regularization terms are added to the log-likelihood function.

### 2.3 Logistic regression

We use a logistic Rregression (LR) formalism as it is particularly convenient when imposing additional statistical properties on the matrices $\mathbf{U}, \mathbf{V}, \mathbf{A}, \mathbf{B}$ such as smoothness or sparseness. In addition, in our experience, LR performs well in strongly overlapping high-dimensional datasets and is insensitive to outliers, the later being of particular concern when including quadratic features.

Under the logistic regression model (2) the class posterior probability $P(y|\mathbf{X};\theta)$ is modeled as

$$P(y|\mathbf{X};\theta) = \frac{1}{1+e^{-y(f(\mathbf{X};\theta)+w_o)}} \qquad (4)$$

and the resulting log likelihood is given by

$$L(\theta) = -\sum_{n=1}^{N} \log(1 + e^{-y(f(\mathbf{X}_n;\theta)+w_o)}) \qquad (5)$$

We minimize the negative log likelihood and add a log-prior on each of the columns of $\mathbf{U}$, $\mathbf{V}$ and $\mathbf{A}$ and parameters of $\mathbf{B}$ that act as a regularization term, which is written as:

$$\underset{\mathbf{U},\mathbf{V},\mathbf{A},\mathbf{B},w_o}{\operatorname{argmin}} \left( -L(\theta) - \sum_{r=1}^{R}(\log p(\mathbf{u}_r) + \log p(\mathbf{v}_r)) - \sum_{k=1}^{K}\log p(\mathbf{a}_k) - \sum_{t=1}^{T'}\log(p(\mathbf{b}_t)) \right) \qquad (6)$$

where the log-priors are given for each of the parameters as $\log p(\mathbf{u}_k) = \mathbf{u}_k^{\mathrm{T}}\mathbf{K}^{(u)}\mathbf{u}_k$, $\log p(\mathbf{v}_k) = \mathbf{u}_k^{\mathrm{T}}\mathbf{K}^{(v)}\mathbf{u}_k$, $\log p(\mathbf{a}_k) = \mathbf{a}_k^{\mathrm{T}}\mathbf{K}^{(a)}\mathbf{a}_k$ and $\log p(\mathbf{b}_k) = \mathbf{b}_k^{\mathrm{T}}\mathbf{K}^{(b)}\mathbf{b}_k$. $\mathbf{K}^{(u)} \in \mathbb{R}^{D\times D}, \mathbf{K}^{(v)} \in \mathbb{R}^{T\times T}, \mathbf{K}^{(a)} \in \mathbb{R}^{D\times D}, \mathbf{K}^{(b)} \in \mathbb{R}^{T\times T}$ are kernel matrices that control the smoothness of the parameter space. Details on the regularization procedure can be found in [8].

Analytic gradients of the log likelihood (5) with respect to the various parameters are given by:

$$\frac{\partial L(\theta)}{\partial \mathbf{u}_r} = \sum_{n=1}^{N} y_n \pi(\mathbf{X}_n)\mathbf{X}_n\mathbf{v}_r \tag{7}$$

$$\frac{\partial L(\theta)}{\partial \mathbf{v}_r} = \sum_{n=1}^{N} y_n \pi(\mathbf{X}_n)\mathbf{u}_r\mathbf{X}_n \tag{8}$$

$$\frac{\partial L(\theta)}{\partial \mathbf{a}_r} = 2\sum_{n=1}^{N} y_n \pi(\mathbf{X}_n)\mathbf{\Lambda}_{r,r}(\mathbf{X}_n\mathbf{B})(\mathbf{X}_n\mathbf{B})^{\mathrm{T}}\mathbf{a}_r \tag{9}$$

$$\frac{\partial L(\theta)}{\partial \mathbf{b}_t} = 2\sum_{n=1}^{N} y_n \pi(\mathbf{X}_n)\mathbf{X}^{\mathrm{T}}\mathbf{A}\mathbf{\Lambda}\mathbf{A}^{\mathrm{T}}\mathbf{X}\mathbf{b}_t \tag{10}$$

where we define

$$\pi(\mathbf{X}_n) = 1 - P(y|\mathbf{X}) = \frac{e^{-y(f(\mathbf{X}_n;\theta)+w_o)}}{1+e^{-y(f(\mathbf{X}_n;\theta)+w_o)}} \tag{11}$$

where $\mathbf{u}_i, \mathbf{v}_i, \mathbf{a}_i,$ and $\mathbf{b}_i$ correspond to the $i_{th}$ columns of $\mathbf{U}, \mathbf{V}, \mathbf{A}, \mathbf{B}$ respectively.

### 2.4 Fourier Basis for $\mathbf{B}$

If matrix $\mathbf{B}$ is constrained to have a circular toepliz structure then it can be represented as $\mathbf{B} = \mathbf{F}^{-1}\mathbf{DF}$, where $\mathbf{F}^{-1}$ denotes the inverse Fourier matrix, and $\mathbf{D}$ is a diagonal complex-valued matrix of Fourier coefficients. In such a case, we can re-write equations (9) and (10) as

$$\frac{\partial L(\theta)}{\partial \mathbf{a}_r} = 2\sum_{n=1}^{N} y_n \pi(\mathbf{X}_n)\mathbf{\Lambda}_{r,r}(\mathbf{X}_n\mathbf{F}^{-1}\hat{\mathbf{D}}\mathbf{F}^{-\mathrm{T}}\mathbf{X}_n^{\mathrm{T}})\mathbf{a}_r \tag{12}$$

$$\frac{\partial L(\theta)}{\partial d_i} = 2\sum_{n=1}^{N} y_n \pi(\mathbf{X}_n)(\mathbf{F}^{-\mathrm{T}}\mathbf{X}_n^{\mathrm{T}}\mathbf{A}\mathbf{\Lambda}\mathbf{A}^{\mathrm{T}}\mathbf{X}_n\mathbf{F}^{-1})_{i,i}d_i \tag{13}$$

$$\tag{14}$$

where $\hat{\mathbf{D}} = \mathbf{D}\mathbf{D}^{\mathrm{T}}$, and the parameters are now optimized with respect to Fourier coefficients $d_i = \mathbf{D}_{i,i}$. An iterative minimization procedure can be used to solve the above minimization.

## 3 Results

### 3.1 Simulated data

In order to validate our method and its ability to capture both linear and second order features, we generated simulated data that contained both types of features; namely ERP type of features and ERS/ERD type of features. The simulated signals were generated with a signal to noise ratio of $-20dB$ which is a typical noise level for EEG. A total of 28 channels, 500 ms long signals and at a sampling frequency of 100Hz where generated, resulting in a matrix of $\mathbf{X}$ of 28 by 50 elements, for each trial. Data corresponding to a total of 1000 trials were generated; 500 trials contained only zero mean Gaussian noise (representing baseline conditions), with the other 500 trials having the signal of interest added to the noise (representing the stimulus condition): For channels 1-9 the signal was composed of a 10Hz sinusoid with random phase in each of the nine channels, and across trials. The

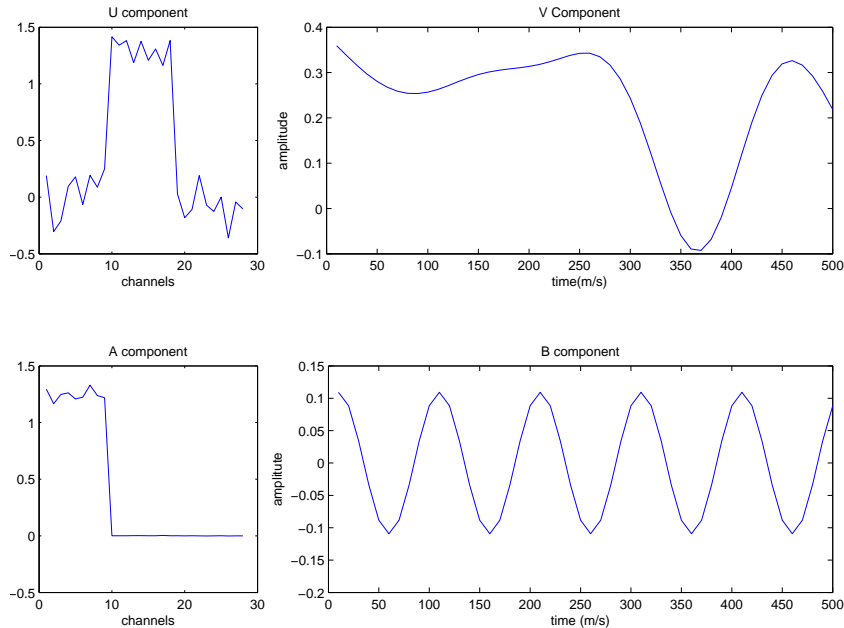

Figure 1: Spatial and temporal component extracted on simulated data for the linear term (top) and quadratic term (bottom).

sinusoids were scaled to match the $-20dB$ SNR. This simulates an ERS type feature. For channels 10-18, a peak represented by a half cycle sinusoid was added at approximately 400 ms, which T simulates an ERP type feature.

The extracted components are shown in Figure 1. The linear component $\mathbf{U}$ (in this case only a column vector) has non-zero coefficients for channels 10 to 18 only, showing that the method correctly identified the ERP activity. Furthermore, the associated temporal component $\mathbf{V}$ has a temporal profile that matches the time course of the simulated evoked response. Similarly, the second order components $\mathbf{A}$ have non-zero weights for only channels 1-9 showing that the method also identified the spatial distribution of the non-phase locked activity. The temporal filter $\mathbf{B}$ was trained in the frequency domain and the resulting filter is shown here in the time domain. It exhibits a dominant 10Hz component, which is indeed the frequency of the non-phase locked activity.

## 3.2 BCI competition dataset

To evaluate the performance of the proposed method on real data we applied the algorithm to an EEG data set that was made available through The BCI Competition 2003 ([14], Data Set IV). EEG was recorded on 28 channels for a single subject performing self-paced key typing, that is, pressing the corresponding keys with the index and little fingers in a self-chosen order and timing (i.e. self-paced). Key-presses occurred at an average speed of 1 key per second. Trial matrices were extracted by epoching the data starting 630ms before each key-press. A total of 416 epochs were recorded, each of length 500ms. For the competition, the first 316 epochs were to be used for classifier training, while the remaining 100 epochs were to be used as a test set. Data were recorded at 1000 Hz with a pass-band between 0.05 and 200 Hz, then downsampled to 100Hz sampling rate.

For this experiment, the matrix $\mathbf{B}$ was fixed to a Toeplitz structure that encodes a 10Hz-33Hz bandpass filter and only the parameters $\mathbf{U}, \mathbf{V}, \mathbf{A}$ and $w_0$ were trained. The number of columns of $\mathbf{U}$ and $\mathbf{V}$ were set to 1, where two columns were used for $\mathbf{A}$. The temporal filter was selected based on prior knowledge of the relevant frequency band. This demonstrates the flexibility of our approach to either incorporate prior knowledge when available or extract it from

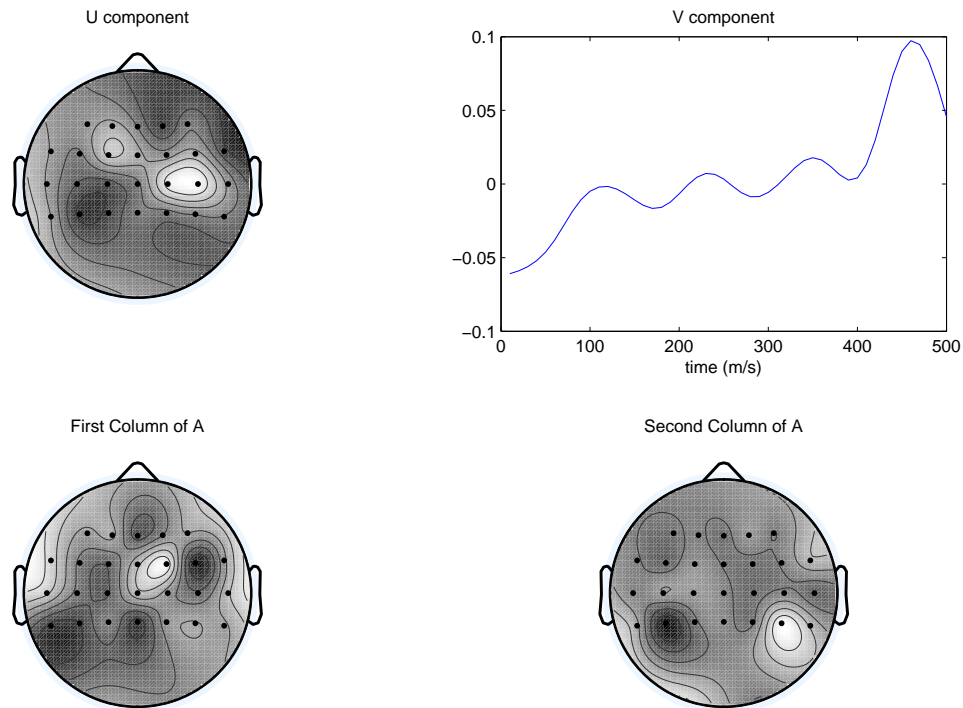

Figure 2: Spatial and temporal component (top), and two spatial components for second order features (bottom) learned on the benchmark dataset

data otherwise. Regularization parameters where chosen via a five fold cross validation procedure (details can be found in [8]). The resulting components for this dataset are shown in Figure 2.

Benchmark performance was measured on the test set which had not been used during either training or cross validation. The number of misclassified trials in the test set was 13 which places our method on a new first place given the results of the competition which can be found online http://ida.first.fraunhofer.de/projects/bci/competition_ii/results/index.html ([14]). Hence, our method works as a classifier producing a state-of-the art result on a realistic data set. The receiver-operator characteristic (ROC) curve for cross validation and for the independent testset are shown in Figure 3. Figure 3.2 also shows the contribution of the linear and quadratic terms for every trial for the two types of key-presses. The figure shows that the two terms provide independent information and that in this case the optimal relative weighting factor is $C \approx 0.5$.

## 4  Conclusion

In this paper we have presented a framework for uncovering spatial as well as temporal features in EEG that combine the two predominant paradigms used in EEG analysis: event related potentials and oscillatory power. These represent phase locked activity (where polarity of the activity matters), and non-phase locked activity (where only the power of the signal is relevant). We used the probabilistic formalism of logistic regression that readily incorporates prior probabilities to regularize the increased number of parameters. We have evaluated the proposed method on both simulated data, and a real BCI benchmark dataset, achieving state-of-the-art classification performance.

The proposed method provides a basis for various future directions. For example, different sets of basis functions (other than a Fourier basis) can be enforced on the temporal decomposition of the data through the matrix $\mathbf{B}$ (e.g. wavelet basis). Further, the method can be easily generalized to

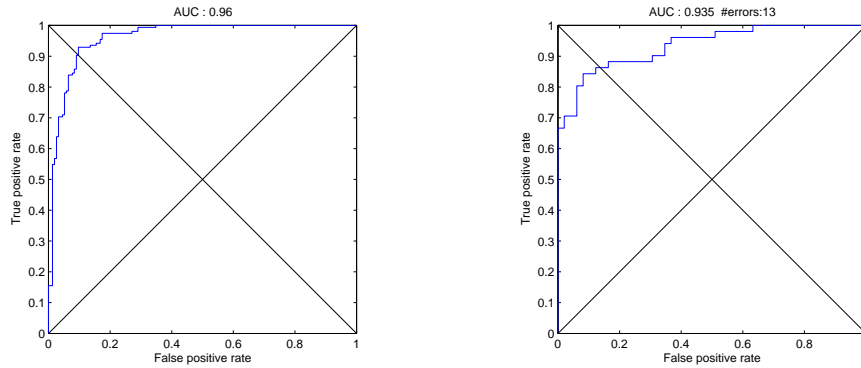

Figure 3: ROC curve with area under the curve 0.96 for the cross validation on the benchmark dataset (left). ROC curve with area under the curve 0.93, on the independent test set, for the benchmark dataset. There were a total of 13 errors on unseen data, which is less than any of the results previously reported, placing this method in first place in the benchmark ranking.

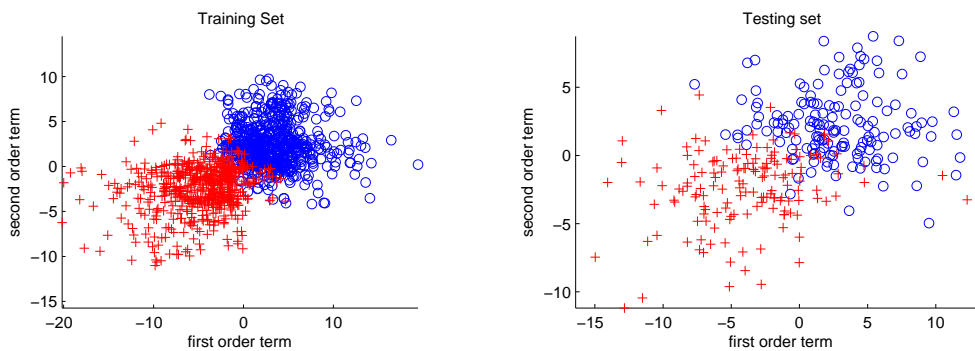

Figure 4: Scatter plot of the first order term vs second order term of the model, on the training and testing set for the benchmark dataset ('+' left key, and 'o' right key). It is clear that the two types of features contain independent information that can help improve the classification performance.

multi-class problems by using a multinomial distribution on $y$. Finally, different regularizations (i.e $L_1$ norm, $L_2$ norm) can be applied to the different types of parameters of the model.

## References

[1] J. R. Wolpaw, N. Birbaumer, D. J. McFarland, G. Pfurtscheller, and T. M. Vaughan. Brain-computer interfaces for communication and control. *Clin Neurophysiol*, 113(6):767–791, June 2002.

[2] N. Birbaumer, N. Ghanayim, T. Hinterberger, I. Iversen, B. Kotchoubey, A. Kubler, J. Perelmouter, E. Taub, and H. Flor. A spelling device for the paralysed. *Nature*, 398(6725):297–8, Mar February-May 1999.

[3] B. Blankertz, G. Curio, and K. uller. Classifying single trial eeg: Towards brain computer interfacing. In T. G. Diettrich, S. Becker, and Z. Ghahramani, editors, *Advances in Neural Information Processing Systems 14*. MIT Press, 2002., 2002.

[4] B. Blankertz, G. Dornhege, C. Schfer, R. Krepki, J. Kohlmorgen, K. Mller, V. Kunzmann, F. Losch, and G. Curio. Boosting bit rates and error detection for the classification of fast-paced motor commands based on single-trial eeg analysis. *IEEE Trans. Neural Sys. Rehab. Eng.*, 11(2):127–131, 2003.

[5] Adam D. Gerson, Lucas C. Parra, and Paul Sajda. Cortically-coupled computer vision for rapid image search. *IEEE Transactions on Neural Systems and Rehabilitation Engineering*, 14:174–179, June 2006.

[6] Lucas C. Parra, Christoforos Christoforou, Adam D. Gerson, Mads Dyrholm, An Luo, Mark Wagner, Marios G. Philiastides, and Paul Sajda. Spatiotemporal linear decoding of brain state: Application to performance augmentation in high-throughput tasks. *IEEE, Signal Processing Magazine*, January 2008.

[7] Philiastides Marios G., Ratcliff Roger, and Sajda Paul. Neural representation of task difficulty and decision making during perceptual categorization: A timing diagram. *Journal of Neuroscience*, 26(35): 8965–8975, August 2006.

[8] Mads Dyrholm, Christoforos Christoforou, and Lucas C. Parra. Bilinear discriminant component analysis. *J. Mach. Learn. Res.*, 8:1097–1111, 2007.

[9] Ryota Tomioka and Kazuyuki Aihara. Classifying matrices with a spectral regularization. In *24th International Conference on Machine Learning*, 2007.

[10] H. Ramoser, J. Müller-Gerking, and G. Pfurtscheller. Optimal spatial filtering of single trial EEG during imagined hand movement. *IEEE Trans. Rehab. Eng.*, 8:441–446, December 2000.

[11] Ryota Tomioka, Kazuyuki Aihara, and Klaus-Robert Mller. Logistic regression for single trial eeg classification. In B. Schölkopf, J. Platt, and T. Hoffman, editors, *Advances in Neural Information Processing Systems 19*, pages 1377–1384. MIT Press, Cambridge, MA, 2007.

[12] S. Lemm, B. Blankertz, G. Curio, and K. Muller. Spatio-spectral filters for improving the classification of single trial eeg. *IEEE Trans Biomed Eng., 52(9):1541–8, 2005.*, 2005.

[13] Dornhege G., Blankertz B, and K.R. Krauledat M. Losch F. Curio G.Muller. Combined optimization of spatial and temporal filters for improving brain-computer interfacing. *IEEE Trans. Biomed. Eng. 2006*, 2006.

[14] B. Blankertz, K.-R. Muller, G. Curio, T.M. Vaughan, G. Schalk, J.R. Wolpaw, A. Schlogl, C. Neuper, G. Pfurtscheller, T. Hinterberger, M. Schroder, and N. Birbaumer. The bci competition 2003: progress and perspectives in detection and discrimination of eeg single trials. *Biomedical Engineering, IEEE Transactions on*, 51(6):1044–1051, 2004.

